# Kernel Latent SVM for Visual Recognition

**Weilong Yang**
School of Computing Science
Simon Fraser University
wya16@sfu.ca

**Yang Wang**
Department of Computer Science
University of Manitoba
ywang@cs.umanitoba.ca

**Arash Vahdat**
School of Computing Science
Simon Fraser University
avahdat@sfu.ca

**Greg Mori**
School of Computing Science
Simon Fraser University
mori@cs.sfu.ca

## Abstract

Latent SVMs (LSVMs) are a class of powerful tools that have been successfully applied to many applications in computer vision. However, a limitation of LSVMs is that they rely on linear models. For many computer vision tasks, linear models are suboptimal and nonlinear models learned with kernels typically perform much better. Therefore it is desirable to develop the kernel version of LSVM. In this paper, we propose *kernel latent SVM* (KLSVM) – a new learning framework that combines latent SVMs and kernel methods. We develop an iterative training algorithm to learn the model parameters. We demonstrate the effectiveness of KLSVM using three different applications in visual recognition. Our KLSVM formulation is very general and can be applied to solve a wide range of applications in computer vision and machine learning.

## 1 Introduction

We consider the problem of learning discriminative classification models for visual recognition. In particular, we are interested in models that have the following two characteristics: 1) can be used on weakly labeled data; 2) have nonlinear decision boundaries.

Linear classifiers are a class of popular learning methods in computer vision. In the case of binary classification, they are prediction models in the form of $f(x) = w^\top x$, where $x$ is the feature vector, and $w$ is a vector of model parameters[1]. The classification decision is based on the value of $f(x)$. Linear classifiers are amenable to efficient and scalable learning/inference – an important factor in many computer vision applications that involve high dimension features and large datasets. The person detection algorithm in [2] is an example of the success of linear classifiers in computer vision. The detector is trained by learning a linear support vector machine based on HOG descriptors of positive and negative examples. The model parameter $w$ in this detector can be thought as a statistical template for HOG descriptors of persons.

The reliance on a rigid template $w$ is a major limitation of linear classifiers. As a result, the learned models usually cannot effectively capture all the variations (shape, appearance, pose, etc.) in natural images. For example, the detector in [2] usually only works well when a person is in an upright posture.

In the literature, there are two main approaches for addressing this limitation. The first one is to introduce latent variables into the linear model. In computer vision, this is best exemplified by the success of deformable part models (DPM) [5] for object detection. DPM captures shape and pose variations of an object class with a root template covering the whole object and several part templates. By allowing these parts to deform from their ideal locations with respect to the root template, DPM provides more flexibility than a rigid template. Learning a DPM involves solving a *latent*

*SVM* (LSVM) [5, 17] – an extension of regular linear SVM for handling latent variables. LSVM provides a general framework for handling "weakly labeled data" arising in many applications. For example, in object detection, the training data are weakly labeled because we are only given the bounding boxes of the objects without the detailed annotation for each part. In addition to modeling part deformation, another popular application of LSVM is to use it as a mixture model where the mixture component is represented as a latent variable [5, 6, 16].

The other main approach is to directly learn a nonlinear classifier. The kernel method [1] is a representative example along this line of work. A limitation of kernel methods is that the learning is more expensive than linear classifiers on large datasets, although efficient algorithms exist for certain types of kernels (e.g. histogram intersection kernel (HIK) [10]). One possible way to address the computational issue is to use nonlinear mapping to convert the original feature into some higher dimensional space, then apply linear classifiers in the high dimensional space [14].

Latent SVM and kernel methods represent two different, yet complementary approaches for learning classification models that are more expressive than linear classifiers. They both have their own advantages and limitations. The advantage of LSVM is that it provides a general and elegant formulation for dealing with many weakly supervised problems in computer vision. The latent variables in LSVM can often have some intuitive and semantic meanings. As a result, it is usually easy to adapt LSVM to capture various prior knowledge about the unobserved variables in various applications. Examples of latent variables in the literature include part locations in object detection [5], subcategories in video annotation [16], object localization in image classification [8], etc. However, LSVM is essentially a parametric model. So the capacity of these types of models is limited by the parametric form. In contrast, kernel methods are non-parametric models. The model complexity is implicitly determined by the number of support vectors. Since the number of support vectors can vary depending on the training data, kernel methods can adapt their model complexity to fit the data.

In this paper, we propose *kernel latent SVM (KLSVM)* – a new learning framework that combines latent SVMs and kernel methods. As a result, KLSVM has the benefits of both approaches. On one hand, the latent variables in KLSVM can be something intuitive and semantically meaningful. On the other hand, KLSVM is nonparametric in nature, since the decision boundary is defined implicitly by support vectors. We demonstrate KLSVM on three applications in visual recognition: 1) object classification with latent localization; 2) object classification with latent subcategories; 3) recognition of object interactions.

## 2 Preliminaries

In this section, we introduce some background on latent SVM and on the dual form of SVMs used for deriving kernel SVMs. Our proposed model in Sec. 3 will build upon these two ideas.

**Latent SVM:** We assume a data instance is in the form of $(x, h, y)$, where $x$ is the observed variable and $y$ is the class label. Each instance is also associated with a latent variable $h$ that captures some unobserved information about the data. For example, say we want to learn a "car" model from a set of positive images containing cars and a set of negative images without cars. We know there is a car somewhere in a positive image, but we do not know its exact location. In this case, $h$ can be used to represent the unobserved location of the car in the image. In this paper, we consider binary classification for simplicity, i.e. $y \in \{+1, -1\}$. Multi-class classification can be easily converted to binary classification, e.g. using one-vs-all or one-vs-one strategy. To simplify the notation, we also assume the latent variable $h$ takes its value from a discrete set of labels $h \in \mathcal{H}$. However, our formulation is general. We will show how to deal with more complex $h$ in Sec. 3.2 and in one of the experiments (Sec. 4.3).

In latent SVM, the scoring function of sample $x$ is defined as $f_w(x) = \max_h w^\top \phi(x, h)$, where $\phi(x, h)$ is the feature vector defined for the pair of $(x, h)$. For example, in the "car model" example, $\phi(x, h)$ can be a feature vector extracted from the image patch at location $h$ of the image $x$. The objective function of LSVM is defined as $\mathcal{L}(w) = \frac{1}{2}||w||^2 + C \sum_i \max(0, 1 - y_i f_w(x_i))$. LSVM is essentially a non-convex optimization problem. However, the learning problem becomes convex once the latent variable $h$ is fixed for positive examples. Therefore, we can train the LSVM by an iterative algorithm that alternates between inferring $h$ on positive examples and optimizing the model parameter $w$.

**Dual form with fixed $h$ on positive examples :** Due to its nature of non-convexity, it is not straightforward to derive the dual form for the general LSVM. Therefore, as a starting point, we first consider a simpler scenario assuming $h$ is fixed (or observed) on the positive training examples. As previously mentioned, the LSVM is then relaxed to a convex problem with this assumption. Note that we will relax this assumption in Sec. 3. In the above "car model" example, this means that we have the ground-truth bounding boxes of the cars in each image. More formally, we are given

$M$ positive samples $\{x_i, h_i\}_{i=1}^{M}$, and $N$ negative samples $\{x_j\}_{j=M+1}^{M+N}$. Inspired by linear SVMs, our goal is to find a linear discriminant $f_w(x, h) = w^\top \phi(x, h)$ by solving the following quadratic program:

$$\mathcal{P}(w^*) = \min_{w, \xi} \ \frac{1}{2}||w||^2 + C_1 \sum_i \xi_i + C_2 \sum_{j,h} \xi_{j,h} \tag{1a}$$

$$\text{s.t. } w^\top \phi(x_i, h_i) \geq 1 - \xi_i, \ \forall i \in \{1, 2, ..., M\}, \tag{1b}$$

$$-w^\top \phi(x_j, h) \geq 1 - \xi_{j,h} \ \forall j \in \{M+1, M+2, ..., M+N\}, \forall h \in \mathcal{H} \tag{1c}$$

$$\xi_i \geq 0, \ \xi_{j,h} \geq 0 \ \forall i, \ \forall j, \ \forall h \in \mathcal{H} \tag{1d}$$

Similar to standard SVMs, $\{\xi_i\}$ and $\{\xi_{j,h}\}$ are the slack variables for handling soft margins.

It is interesting to note that the optimization problem in Eq. 1 is almost identical to that of standard linear SVMs. The only difference lies in the constraint on the negative training examples (Eq. 1c). Since we assume $h$'s are not observed on negative images, we need to enumerate all possible values for $h$'s in Eq. 1c. Intuitively, this means every image patch from a negative image (i.e. non-car image) is not a car.

It is easy to show that Eq. 1 is convex. Similar to the dual form of standard SVMs, we can derive the dual form of Eq. 1 as follows:

$$\mathcal{D}(\alpha^*, \beta^*) = \max_{\alpha, \beta} \ \sum_i \alpha_i + \sum_j \sum_h \beta_{j,h} - \frac{1}{2}||\sum_i \alpha_i \phi(x_i, h_i) - \sum_j \sum_h \beta_{j,h} \phi(x_j, h)||^2 \tag{2a}$$

$$\text{s.t. } 0 \leq \alpha_i \leq C_1, \ \forall i; \ 0 \leq \beta_{j,h} \leq C_2, \ \forall j, \ \forall h \in \mathcal{H} \tag{2b}$$

The optimal primal parameters $w^*$ for Eq. 1 and the optimal dual parameters $(\alpha^*, \beta^*)$ for Eq. 2 are related as follows:

$$w^* = \sum_i \alpha_i^* \phi(x_i, h_i) - \sum_j \sum_h \beta_{j,h}^* \phi(x_j, h) \tag{3}$$

Let us define $\lambda$ to be the concatenations of $\{\alpha_i : \forall i\}$ and $\{\beta_{j,h} : \forall j, \forall h \in \mathcal{H}\}$, so $|\lambda| = M + N \times |\mathcal{H}|$. Let $\Psi$ be a $|\lambda| \times D$ matrix where $D$ is the dimension of $\phi(x, h)$. $\Psi$ is obtained by stacking together $\{\phi(x_i, h_i) : \forall i\}$ and $\{-\phi(x_j, h) : \forall j, \forall h \in \mathcal{H}\}$. We also define $Q = \Psi\Psi^\top$ and $\mathbb{1}$ to be a vector of all 1's. Then Eq. 2a can be rewritten as (we omit the linear constraints on $\lambda$ for simplicity):

$$\max_\lambda \lambda^\top \cdot \mathbb{1} - \frac{1}{2}\lambda^\top Q\lambda \tag{4}$$

The advantage of working with the dual form in Eq. 4 is that it only involves a so-called kernel matrix $Q$. Each entry of $Q$ is a dot-product of two vectors in the form of $\phi(x, h)^\top \phi(x', h')$. We can replace the dot-product with any other kernel functions in the form of $k(\phi(x, h), \phi(x', h'))$ to get nonlinear classifiers [1]. The scoring function for the testing images $x^{new}$ can be kernelized as follows: $f(x^{new}) = \max_{h^{new}} \left( \sum_i \alpha_i^* k(\phi(x_i, h_i), \phi(x^{new}, h^{new})) - \sum_j \sum_h \beta_{j,h}^* k(\phi(x_j, h), \phi(x^{new}, h^{new})) \right)$.

Another important, yet often overlooked fact is that the optimal values of the two quadratic programs in Eqs. 1 and 2 have some specific meanings. They correspond to the inverse of the (soft) margin of the resultant SVM classifier [9, 15]: $\mathcal{P}(w^*) = \mathcal{D}(\alpha^*, \beta^*) = \frac{1}{\text{SVM margin}}$. In the next section, we will exploit this fact to develop the kernel latent support vector machines.

## 3 Kernel Latent SVM

Now we assume the variables $\{h_i\}_{i=1}^{M}$ on the positive training examples are unobserved. If the scoring function used for classification is in the form of $f(x) = \max_h w^\top \phi(x, h)$, we can use the LSVM formulation [5, 17] to learn the model parameters $w$. As mentioned earlier, the limitation of LSVM is the linearity assumption of $w^\top \phi(x, h)$. In this section, we propose kernel latent SVM (KLSVM) – a new latent variable learning method that only requires a kernel function $K(x, h, x', h')$ between a pair of $(x, h)$ and $(x', h')$.

Note that when $\{h_i\}_{i=1}^{M}$ are observed on the positive training examples, we can plug them in Eq. 2 to learn a nonlinear kernelized decision function that separates the positive and negative examples.

When $\{h_i\}_{i=1}^{M}$ are latent, an intuitive thing to do is to find the labeling of $\{h_i\}_{i=1}^{M}$ so that when we plug them in and solve for Eq. 2, the resultant nonlinear decision function separates the two classes as widely as possible. In other words, we look for a set of $\{h_i^*\}$ which can maximize the SVM margin (equivalent to minimizing $\mathcal{D}(\alpha^*, \beta^*, \{h_i\})$). The same intuition was previously used to develop the max-margin clustering method in [15]. Using this intuition, we write the optimal function value of the dual form as $\mathcal{D}(\alpha^*, \beta^*, \{h_i\})$ since now it implicitly depends on the labelings $\{h_i\}$. We can jointly find the labelings $\{h_i\}$ and solve for $(\alpha^*, \beta^*)$ by the following optimization problem:

$$\min_{\{h_i\}} \mathcal{D}(\alpha^*, \beta^*, \{h_i\}) \tag{5a}$$

$$= \min_{\{h_i\}} \max_{\alpha, \beta} \ \sum_i \alpha_i + \sum_j \sum_h \beta_{j,h} - \frac{1}{2} || \sum_i \alpha_i \phi(x_i, h_i) - \sum_j \sum_h \beta_{j,h} \phi(x_j, h)||^2 \tag{5b}$$

$$\text{s.t. } 0 \le \alpha_i \le C_1, \ \forall i; \quad 0 \le \beta_{j,h} \le C_2, \ \forall j, \ \forall h \in \mathcal{H} \tag{5c}$$

The most straightforward way of solving Eq. 5 is to optimize $\mathcal{D}(\alpha^*, \beta^*, \{h_i\})$ for every possible combination of values for $\{h_i\}$, and then take the minimum. When $h_i$ takes its value from a discrete set of $K$ possible choices (i.e. $|\mathcal{H}| = K$), this naive approach needs to solve $M^K$ quadratic programs. This is obviously too expensive. Instead, we use the following iterative algorithm:

- Fix $\alpha$ and $\beta$, compute the optimal $\{h_i\}^*$ by

$$\{h_i\}^* = \arg\max_{\{h_i\}} \frac{1}{2} || \sum_i \alpha_i \phi(x_i, h_i) - \sum_j \sum_h \beta_{j,h} \phi(x_j, h)||^2 \tag{6}$$

- Fix $\{h_i\}$, compute the optimal $(\alpha^*, \beta^*)$ by

$$(\alpha^*, \beta^*) = \arg\max_{\alpha, \beta} \left\{ \sum_i \alpha_i + \sum_j \sum_h \beta_{j,h} - \frac{1}{2} || \sum_i \alpha_i \phi(x_i, h_i) - \sum_j \sum_h \beta_{j,h} \phi(x_j, h)||^2 \right\} \tag{7}$$

The optimization problem in Eq. 7 is a quadratic program similar to that of a standard dual SVM. As a result, Eq. 7 can be kernelized as Eq. 4 and solved using standard dual solver in regular SVMs. In Sec. 3.1, we describe how to kernelize and solve the optimization problem in Eq. 6.

### 3.1 Optimization over $\{h_i\}$

The complexity of a simple enumeration approach for solving Eq. 6 is again $O(M^K)$, which is clearly too expensive for practical purposes. Instead, we solve it iteratively using an algorithm similar to co-ordinate ascent. Within an iteration, we choose one positive training example $t$. We update $h_t$ while fixing $h_i$ for all $i \ne t$. The optimal $h_t^*$ can be computed as follows:

$$h_t^* = \arg\max_{h_t} ||\alpha_t \phi(x_t, h_t) + \sum_{i:i\ne t} \alpha_i \phi(x_i, h_i) - \sum_j \sum_h \beta_{j,h} \phi(x_j, h)||^2 \tag{8a}$$

$$\Leftrightarrow \arg\max_{h_t} ||\alpha_t \phi(x_t, h_t)||^2 + 2 \left( \sum_{i:i\ne t} \alpha_i \phi(x_i, h_i) - \sum_j \sum_h \beta_{j,h} \phi(x_j, h) \right)^\top \alpha_t \phi(x_t, h_t) \tag{8b}$$

By replacing the dot-product $\phi(x, h)^\top \phi(x', h')$ with a kernel function $k(\phi(x, h), \phi(x', h'))$, we obtain the kernerlized version of Eq. 8(b) as follows

$$h_t^* = \arg\max_{h_t} \ \alpha_t \alpha_t k(\phi(x_t, h_t), \phi(x_t, h_t)) + 2 \sum_{i:i\ne t} \alpha_i \alpha_t k(\phi(x_i, h_i), \phi(x_t, h_t))$$

$$-2 \sum_j \sum_h \beta_{j,h} \alpha_t k(\phi(x_j, h), \phi(x_t, h_t)) \tag{9}$$

It is interesting to notice that if the $t$-th example is not a support vector (i.e. $\alpha_t = 0$), the function value of Eq. 9 will be zero regardless of the value of $h_t$. This means in KLSVM we can improve the training efficiency by only performing Eq. 9 on positive examples corresponding to support vectors. For other positive examples (non-support vectors), we can simply set their latent variables the same

as the previous iteration. Note that in LSVM, the inference during training needs to be performed on every positive example.

**Connection to LSVM:** When a linear kernel is used, the inference problem (Eq. 8) has a very interesting connection to LSVM in [5]. Recall that for linear kernels, the model parameters $w$ and dual variables $(\alpha, \beta)$ are related by Eq. 3. Then Eq. 8 becomes:

$$h_t^* = \arg\max_{h_t} ||\alpha_t \phi(x_t, h_t)||^2 + 2\left(w - \alpha_t \phi(x_t, h_t^{\text{old}})\right)^\top \alpha_t \phi(x_t, h_t) \tag{10a}$$

$$\Leftrightarrow \arg\max_{h_t} \alpha_t w^\top \phi(x_t, h_t) + \frac{1}{2}\alpha_t^2 ||\phi(x_t, h_t)||^2 - \alpha_t^2 \phi(x_t, h_t^{\text{old}})^\top \phi(x_t, h_t) \tag{10b}$$

where $h_t^{\text{old}}$ is the value of latent variable of the $t$-th example in the previous iteration. Let us consider the situation when $\alpha_t \neq 0$ and the feature vector $\phi(x, h)$ is $l_2$ normalized, which is commonly used in computer vision. In this case, $\alpha_t^2 \phi(x_t, h_t)^\top \phi(x_t, h_t)$ is a constant, and we have $\phi(x_t, h_t^{\text{old}})^\top \phi(x_t, h_t^{\text{old}}) > \phi(x_t, h_t^{\text{old}})^\top \phi(x_t, h_t)$ if $h_t \neq h_t^{\text{old}}$. Then Eq. 10 is equivalent to:

$$h_t^* = \arg\max_{h_t} w^\top \phi(x_t, h_t) - \alpha_t \phi(x_t, h_t^{\text{old}})^\top \phi(x_t, h_t) \tag{11}$$

Eq. 11 is very similar to the inference problem in LSVM, i.e., $h_t^* = \arg\max_{h_t} w^\top \phi(x_t, h_t)$, but with an extra term $\alpha_t \phi(x_t, h_t^{\text{old}})^\top \phi(x_t, h_t)$ which penalizes the choice of $h_t$ for being the same value as previous iteration $h_t^{\text{old}}$. This has a very appealing intuitive interpretation. If the $t$-th positive example is a support vector, the latent variable $h^{old}$ from previous iteration causes this example to lie very close to (or even on the wrong side) the decision boundary, i.e. the example is not well-separated. During the current iteration, the second term in Eq. 11 penalizes $h^{old}$ to be chosen again since we already know the example will not be well-separated if we choose $h^{old}$ again. The amount of penalty depends on the magnitudes of $\alpha_t$ and $\phi(x_t, h_t^{old})^\top \phi(x_t, h_t)$. We can interpret $\alpha_t$ as how "bad" $h_t^{old}$ is, and $\phi(x_t, h_t^{old})^\top \phi(x_t, h_t)$ as how close $h_t$ is to $h_t^{old}$. Eq. 11 penalizes the new $h_t^*$ to be "close" to "bad" $h_t^{old}$.

### 3.2 Composite Kernels

So far we have assumed that the latent variable $h$ takes its value from a discrete set of labels. Given a pair of $(x, h)$ and $(x', h')$, the types of kernel function $k(x, h; x', h')$ we can choose from are still limited to a handful of standard kernels (e.g. Gaussian, RBF, HIK, etc). In this section, we consider more interesting cases where $h$ involves some complex structures. This will give us two important benefits. First of all, it allows us to exploit structural information in the latent variables. This is in analog to structured output learning (e.g. [12, 13]). More importantly, it gives us more flexibility to construct new kernel functions by composing from simple kernels.

Before we proceed, let us first motivate the composite kernel with an example application. Suppose we want to detect some complex person-object interaction (e.g. "person riding a bike") in an image. One possible solution is to detect persons and bikes in an image, then combine the results by taking into account of their relationship (i.e. "riding"). Imagine we already have kernel functions corresponding to some components (e.g. person, bike) of the interaction. In the following, we will show how to compose a new kernel for the "person riding a bike" classifier from those components.

We denote the latent variable using $\vec{h}$ to emphasize that now it is a vector instead of a single discrete value. We denote it as $\vec{h} = (z_1, z_2, ...)$, where $z_u$ is the $u$-th component of $\vec{h}$ and takes its value from a discrete set of possible labels. For the structured latent variable, it is assumed that there are certain dependencies between some pairs of $(z_u, z_v)$. We can use an undirected graph $\mathcal{G} = (\mathcal{V}, \mathcal{E})$ to capture the structure of the latent variable, where a vertex $u \in \mathcal{V}$ corresponds to the label $z_u$, and an edge $(u, v) \in \mathcal{E}$ corresponds to the dependency between $z_u$ and $z_v$. As a concrete example, consider the "person riding a bike" recognition problem. The latent variable in this case has two components $\vec{h} = (z_{person}, z_{bike})$ corresponding to the location of person and bike, respectively. On the training data, we have access to the ground-truth bounding box of "person riding a bike" as a whole, but not the exact location of "person" or "bike" within the bounding box. So $\vec{h}$ is latent in this application. The edge connecting $z_{person}$ and $z_{bike}$ captures the relationship (e.g. "riding on", "next to", etc.) between these two objects.

Suppose we already have kernel functions corresponding to the vertices and edges in the graph, we can then define the composite kernel as the summation of the kernels over all the vertices and edges.

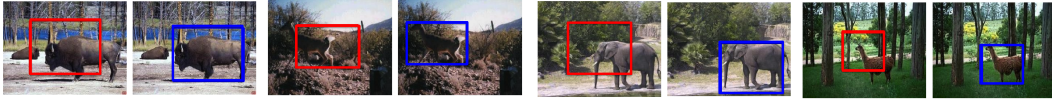

Figure 1: Visualization of how the latent variable (i.e. object location) changes during the learning. The red bounding box corresponds to the initial object location. The blue bounding box corresponds to the object location after the learning.

| Method | BOF + linear SVM | BOF + kernel SVM | linear LSVM | KLSVM |
|---|---|---|---|---|
| Acc (%) | $45.57 \pm 4.23$ | $50.53 \pm 6.53$ | $75.07 \pm 4.18$ | $\mathbf{84.49 \pm 3.63}$ |

Table 1: Results on the mammal dataset. We show the mean/std of classification accuracies over five rounds of experiments.

$$K(\Phi(x, \vec{h}), \Phi(x', \vec{h}')) = \sum_{u \in \mathcal{V}} k_u(\phi(x, z_u), \phi(x', z'_u)) + \sum_{(u,v) \in \mathcal{E}} k_{uv}(\psi(x, z_u, z_v), \psi(x', z'_u, z'_v)) \quad (12)$$

When the latent variable $\vec{h}$ forms a tree structure, there exist efficient inference algorithms for solving Eq. 9, such as dynamic programming. It is also possible for Eq. 12 to include kernels defined on higher-order cliques in the graph, as long as we have some pre-defined kernel functions for them.

## 4 Experiments

We evaluate KLSVM in three different applications of visual recognition. Each application has a different type of latent variables. For these applications, we will show that KLSVM outperforms both the linear LSVM [5] and the regular kernel SVM. Note that we implement the learning of linear LSVM by ourselves using the same iterative algorithm as the one in [5].

### 4.1 Object Classification with Latent Localization

**Problem and Dataset:** We consider object classification with image-level supervision. Our training data only have image-level labels indicating the presence/absence of each object category in an image. The exact object location in the image is not provided and is considered as the latent variable $h$ in our formulation. We define the feature vector $\phi(x, h)$ as the HOG feature extracted from the image at location $h$. During testing, the inference of $h$ is performed by enumerating all possible locations of the image.

We evaluate our algorithm on the mammal dataset [8] which consists of 6 mammal categories. There are about 45 images per category. For each category, we use half of the images for training and the remaining half for testing. We assume the object size is the same for the images of the same category, which is a reasonable assumption for this dataset. This dataset was used to evaluate the linear LSVM in [8].

**Results:** We compare our algorithm with linear LSVM. To demonstrate the benefit of using latent variables, we also compare with two simple baselines using linear and kernel SVMs based on bag-of-features (BOF) extracted from the whole image (i.e. without latent variables). For both baselines, we aggregate the quantized HOG features densely sampled from the whole image. Then, the features are fed into the standard linear SVM and kernel SVM respectively. We use the histogram intersection kernel (HIK) [10] since it has been proved to be successful for vision applications, and efficient learning/inference algorithms exist for this kernel.

We run the experiments for five rounds. In each round, we randomly split the images from each category into training and testing sets. For both linear LSVM and KLSVM, we initialize the latent variable at the center location of each image and we set $C_1 = C_2 = 1$. For both algorithms, we use one-versus-one classification scheme. We use the HIK kernel in the KLSVM. Table 1 summarizes the mean and standard deviations of the classification accuracies over five rounds of experiments. Across all experiments, both linear LSVM and KLSVM achieve significantly better results than approaches using BOF features from the whole image. This is intuitively reasonable since most of images on this dataset share very similar scenes. So BOF feature without latent variables cannot capture the subtle differences between each category. Table 1 also shows KLSVM significantly outperforms linear LSVM.

Fig. 1 shows examples of how the latent variables change on some training images during the learning of the KLSVM. For each training image, the location of the object (latent variable $h$) is initialized to the center of the image. After the learning algorithm terminates, the latent variables accurately locate the objects.

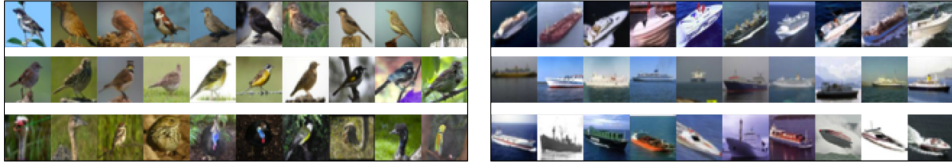

Figure 2: Visualization of some testing examples from the "bird" (left) and "boat" (right) categories. Each row corresponds to a subcategory. We can see that visually similar images are grouped into the same subcategory.

| Method | non-latent linear SVM | linear LSVM | non-latent kernel SVM | KLSVM |
|---|---|---|---|---|
| Acc (%) | $50.69 \pm 0.38$ | $53.13 \pm 0.63$ | $52.98 \pm 0.22$ | $\mathbf{55.17 \pm 0.27}$ |

Table 2: Results on CIFAR10 Dataset. We show the mean/std of classification accuracies over five folds of experiments. Each fold uses a different batch of the training data.

## 4.2 Object Classification with Latent Subcategory

**Problem and Dataset:** Our second application is also on object classification. But here we consider a different type of latent variable. Objects within a category usually have a lot of intra-class variations. For example, consider the images for the "bird" category shown in the left column of Fig. 2. Even though they are examples of the same category, they still exhibit very large appearance variations. It is usually very difficult to learn a single "bird" model that captures all those variations. One way to handle the intra-class variation is to split the "bird" category into several subcategories. Examples within a subcategory will be more visually similar than across all subcategories. Here we use the latent variable $h$ to indicate the subcategory an image belongs to. If a training image belongs to the class $c$, its subcategory label $h$ takes value from a set $\mathcal{H}_c$ of subcategory labels corresponding to the $c$-th class. Note that subcategories are latent on the training data, so they may or may not have semantic meanings.

The feature vector $\phi(x, h)$ is defined as a sparse vector whose feature dimension is $|\mathcal{H}_c|$ times of the dimension of $\phi(x)$, where $\phi(x)$ is the HOG descriptor extracted from the image $x$. In the experiments, we set $|\mathcal{H}_c| = 3$ for all $c$'s. Then we can define $\phi(x, h = 1) = (\phi(x); \mathbf{0}; \mathbf{0})$, $\phi(x, h = 2) = (\mathbf{0}; \phi(x); \mathbf{0})$, and so on. Similar models have been proposed to address the viewpoint changing in object detection [6] and semantic variations in YouTube video tagging [16].

We use the CIFAR10 [7] dataset in our experiment. It consists of images from ten classes including airplane, automobile, bird, cat, etc. The training set has been divided into five batches and each batch contains 10000 images. There are in total 10000 test images.

**Results:** Again we compare with three baselines: linear LSVM, non-latent linear SVM, non-latent kernel SVM. Similarly, we use HIK kernel for the kernel-based methods. For non-latent approaches, we simply feed feature vector $\phi(x)$ to SVMs without using any latent variable.

We run the experiments in five folds. Each fold use a different training batch but the same testing batch. We set $C_1 = C_2 = 0.01$ for all the experiments and initialize the subcategory labels of training images by k-means clustering. Table 2 summarizes the results. Again, KLSVM outperforms other baseline approaches. It is interesting to note that both linear LSVM and KLSVM outperform their non-latent counterparts, which demonstrates the effectiveness of using latent subcategories in object classification. We visualize examples of the correctly classified testing images from the "bird" and "boat" categories in Fig. 2. Images on the same row are assigned the same subcategory labels. We can see that visually similar images are automatically grouped into the same subcategory.

## 4.3 Recognition of Object Interaction

**Problem and Dataset:** Finally, we consider an application where the latent variable is more complex and requires the composite kernel introduced in Sec. 3.2. We would like to recognize complex interactions between two objects (also called "visual phrases" [11]) in static images. We build a dataset consisting of four object interaction classes, i.e. "person riding a bicycle", "person next to a bicycle", "person next to a car" and "bicycle next to a car" based on the visual phrase dataset in [11]. Each class contains 86~116 images. Each image is only associated with one of the four object interaction label. There is no ground-truth bounding box information for each object. We use 40 images from each class for training and the rest for testing.

**Our approach:** We treat the locations of objects as latent variables. For example, when learning the model for "person riding a bicycle", we treat the locations of "person" and "bicycle" as latent variables. In this example, each image is associated with latent variables $\vec{h} = (z_1, z_2)$, where $z_1$ denotes the location of the "person" and $z_2$ denotes the location of the "bicycle". To reduce the search space of inference, we first apply off-the-shelf "person" and "bicycle" detectors [5] on

| Method | BOF + linear SVM | BOF + kernel SVM | linear LSVM | KLSVM |
|---|---|---|---|---|
| Acc(%) | 42.92 | 58.46 | $46.33 \pm 1.4$ | $\mathbf{66.42 \pm 0.99}$ |

Table 3: Results on object interaction dataset. For the approaches using latent variables, we show the mean/std of classification accuracies over five folds of experiments.

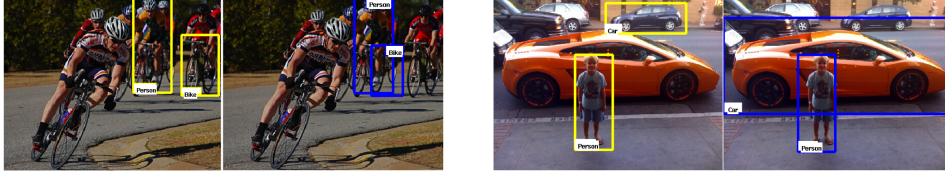

Figure 3: Visualization of how latent variables (i.e. object locations) change during the learning. The left image is from the "person riding a bicycle" category, and the right image is from the "person next to a car" category. Yellow bounding boxes corresponds to the initial object locations. The blue bounding boxes correspond to the object locations after the learning.

each image. For each object, we generate five candidate bounding boxes which form a set $\mathcal{Z}_i$, i.e. $|\mathcal{Z}_1| = |\mathcal{Z}_2| = 5$ and $z_i \in \mathcal{Z}_i$. Then, the inference of $\vec{h}$ is performed by enumerating 25 combinations of $z_1$ and $z_2$. We also assume there are certain dependencies between the pair of $(z_1, z_2)$. Then the kernel between two images can be defined as follows:

$$K(\Phi(x, \vec{h}), \Phi(x', \vec{h}')) = \sum_{u=\{1,2\}} k_u \left( \phi(x, z_u), \phi(x', z'_u) \right) + k_p \left( \psi(z_1, z_2), \psi(z'_1, z'_2) \right) \qquad (13)$$

We define $\phi(x, z_u)$ as the bag-of-features (BOF) extracted from the bounding box $z_u$ in the image $x$. For each bounding box, we split the region uniformly into four equal quadrants. Then we compute the bag-of-features for each quadrant by aggregating quantized HOG features. The final feature vector is the concatenation of these four bag-of-features histograms. This feature representation is similar to the spatial pyramid feature representation. In our experiment, we choose HIK for $k_u(\cdot)$. The kernel $k_p(\cdot)$ captures the spatial relationship between $z_1$ and $z_2$ such as above, below, overlapping, next-to, near, and far. Here $\psi(z_1, z_2)$ is a sparse binary vector and its $k$-th element is set to 1 if the corresponding $k$-th relation is satisfied between bounding boxes $z_1$ and $z_2$. Note that $k_p(\cdot)$ does not depend on the images. Similar representation has been used in [4]. We define $k_p(\cdot)$ as a simple linear kernel.

**Results:** We compare with the simple BOF + linear SVM, and BOF + kernel SVM approaches. These two baselines use the same BOF feature representation as our approach except that the features are extracted from the whole image. We choose the HIK in the kernel SVM. Note that this is a strong baseline since [3] has shown that a similar pyramid feature representation with kernel SVM achieves top performances on the task of person-object interaction recognition. The other baseline is the standard linear LSVM, in which we build the feature vector $\phi(x, h)$ by simply concatenating both unary features and pairwise features, i.e. $\phi(x, h) = [\phi(x, z_1); \phi(x, z_2); \psi(z_1, z_2)]$. Again, we set $C_1 = C_2 = 1$ for all experiments. We run the experiments for five rounds for approaches using latent variables. In each round, we randomly initialize the choices of $z_1$ and $z_2$. Table 3 summarizes the results. The kernel latent SVM that uses HIK for $k_u(\cdot)$ achieves the best performance.

Fig. 3 shows examples of how the latent variables change on some training images during the learning of the KLSVM. For each training image, both latent variables $z_1$ and $z_2$ are randomly initialized to one of five candidate bounding boxes. As we can see, the initial bounding boxes can accurately locate the target objects but their spatial relations are different to ground-truth labels. After learning algorithm terminates, the latent variables not only locate the target objects, but more importantly they also capture the correct spatial relationship between objects.

## 5 Conclusion

We have proposed kernel latent SVM – a new learning framework that combines the benefits of LSVM and kernel methods. Our learning framework is very general. The latent variables can not only be a single discrete value, but also be more complex values with interdependent structures. Our experimental results on three different applications in visual recognition demonstrate that KLSVM outperforms using LSVM or using kernel methods alone. We believe our work will open the possibility of constructing more powerful and expressive prediction models for visual recognition.

**Acknowledgement:** This work was supported by a Google Research Award and NSERC. Yang Wang was partially supported by a NSERC postdoc fellowship.

## Footnotes

[1]Without loss of generality, we assume linear models without the bias term.

# References

[1] C. J. Burges. A tutorial on support vector machines for pattern recognition. *Data Mining and Knowledge Discovery*, 2(2):121–167, 1998.

[2] N. Dalal and B. Triggs. Histogram of oriented gradients for human detection. In *IEEE Computer Society Conference on Computer Vision and Pattern Recognition*, 2005.

[3] V. Delaitre, I. Laptev, and J. Sivic. Recognizing human actions in still images: a study of bag-of-features and part-based representations. In *British Machine Vision Conference*, 2010.

[4] C. Desai, D. Ramanan, and C. Fowlkes. Discriminative models for multi-class object layout. In *IEEE International Conference on Computer Vision*, 2009.

[5] P. F. Felzenszwalb, R. B. Girshick, D. McAllester, and D. Ramanan. Object detection with discriminatively trained part based models. *IEEE Transactions on Pattern Analysis and Machine Intelligence*, 32(9):1672–1645, 2010.

[6] C. Gu and X. Ren. Discriminative mixture-of-templates for viewpoint classification. In *European Conference on Computer Vision*, 2010.

[7] A. Krizhevsky. Learning multiple layers of features from tiny images. Master's thesis, University of Toronto, 2009.

[8] M. P. Kumar, B. Packer, and D. Koller. Self-paced learning for latent variable models. In *Advances in Neural Information Processing Systems*, 2010.

[9] G. R. G. Lanckriet, N. Cristianini, P. Bartlett, L. R. Ghaoui, and M. I. Jordan. Learning the kernel matrix with semidefinite programming. *Journal of Machine Learning Research*, 5:24–72, 2004.

[10] S. Maji, A. C. Berg, and J. Malik. Classification using intersection kernel support vector machines is efficient. In *IEEE Computer Society Conference on Computer Vision and Pattern Recognition*, 2008.

[11] M. A. Sadeghi and A. Farhadi. Recognition using visual phrases. In *IEEE Computer Society Conference on Computer Vision and Pattern Recognition*, 2011.

[12] B. Taskar, C. Guestrin, and D. Koller. Max-margin markov networks. In *Advances in Neural Information Processing Systems*, volume 16. MIT Press, 2004.

[13] I. Tsochantaridis, T. Joachims, T. Hofmann, and Y. Altun. Large margin methods for structured and interdependent output variables. *Journal of Machine Learning Research*, 6:1453–1484, 2005.

[14] A. Vedaldi and A. Zisserman. Efficient additive kernels via explicit feature maps. *Pattern Analysis and Machine Intellingence*, 34(3), 2012.

[15] L. Xu, J. Neufeldand, B. Larson, and D. Schuurmans. Maximum margin clustering. In L. K. Saul, Y. Weiss, and L. Bottou, editors, *Advances in Neural Information Processing Systems*, volume 17, pages 1537–1544. MIT Press, Cambridge, MA, 2005.

[16] W. Yang and G. Toderici. Discriminative tag learning on youtube videos with latent sub-tags. In *IEEE Computer Society Conference on Computer Vision and Pattern Recognition*, 2011.

[17] C.-N. Yu and T. Joachims. Learning structural SVMs with latent variables. In *International Conference on Machine Learning*, 2009.

